# Some Theoretical Results Concerning the Convergence of Compositions of Regularized Linear Functions

**Tong Zhang**
Mathematical Sciences Department
IBM T.J. Watson Research Center
Yorktown Heights, NY 10598
tzhang@watson.ibm.com

## Abstract

Recently, sample complexity bounds have been derived for problems involving linear functions such as neural networks and support vector machines. In this paper, we extend some theoretical results in this area by deriving dimensional independent covering number bounds for regularized linear functions under certain regularization conditions. We show that such bounds lead to a class of new methods for training linear classifiers with similar theoretical advantages of the support vector machine. Furthermore, we also present a theoretical analysis for these new methods from the asymptotic statistical point of view. This technique provides better description for large sample behaviors of these algorithms.

## 1  Introduction

In this paper, we are interested in the generalization performance of linear classifiers obtained from certain algorithms. From computational learning theory point of view, such performance measurements, or sample complexity bounds, can be described by a quantity called covering number [11, 15, 17], which measures the size of a parametric function family. For two-class classification problem, the covering number can be bounded by a combinatorial quantity called VC-dimension [12, 17]. Following this work, researchers have found other combinatorial quantities (dimensions) useful for bounding the covering numbers. Consequently, the concept of VC-dimension has been generalized to deal with more general problems, for example in [15, 11].

Recently, Vapnik introduced the concept of support vector machine [16] which has been successful applied to many real problems. This method achieves good generalization by restricting the 2-norm of the weights of a separating hyperplane. A similar technique has been investigated by Bartlett [3], where the author studied the performance of neural networks when the 1-norm of the weights is bounded. The same idea has also been applied in [13] to explain the effectiveness of the boosting algorithm. In this paper, we will extend their results and emphasize the importance of dimension independence. Specifically, we consider the following form of regularization method (with an emphasis on classification problems) which has been widely studied for regression problems both in statistics and in

numerical mathematics:

$$\inf_w E_{x,y} L(w, x, y) = \inf_w E_{x,y} f(w^T xy) + \lambda g(w), \tag{1}$$

where $E_{x,y}$ is the expectation over a distribution of $(x, y)$, and $y \in \{-1, 1\}$ is the binary label of data vector $x$. To apply this formulation for the purpose of training linear classifiers, we can choose $f$ as a decreasing function, such that $f(\cdot) \geq 0$, and choose $g(w) \geq 0$ as a function that penalizes large $w$ ($\lim_{w \to \infty} g(w) \to \infty$). $\lambda$ is an appropriately chosen positive parameter to balance the two terms.

The paper is organized as follows. In Section 2, we briefly review the concept of covering numbers as well as the main results related to analyzing the performance of learning algorithms. In Section 3, we introduce the regularization idea. Our main goal is to construct regularization conditions so that dimension independent bounds on covering numbers can be obtained. Section 4 extends results from the previous section to nonlinear compositions of linear functions. In Section 5, we give an asymptotic formula for the generalization performance of a learning algorithm, which will then be used to analyze an instance of SVM. Due to the space limitation, we will only present the main results and discuss their implications. The detailed derivations can be found in [18].

## 2  Covering numbers

We formulate the learning problem as to find a parameter from random observations to minimize risk: given a loss function $L(\alpha, x)$ and $n$ observations $X_1^n = \{x_1, \ldots, x_n\}$ independently drawn from a fixed but unknown distribution $D$, we want to find $\alpha$ that minimizes the expected loss over $x$ (*risk*):

$$R(\alpha) = E_x L(\alpha, x) = \int L(\alpha, x) \, dP(x). \tag{2}$$

The most natural method for solving (2) using a limited number of observations is by the *empirical risk minimization* (ERM) method (*cf.* [15, 16]). We simply choose a parameter $\alpha$ that minimizes the observed risk:

$$R(\alpha, X_1^n) = \frac{1}{n} \sum_{i=1}^n L(\alpha, x_i). \tag{3}$$

We denote the parameter obtained in this way as $\alpha_{\text{erm}}(X_1^n)$. The convergence behavior of this method can be analyzed by using the VC theoretical point of view, which relies on the uniform convergence of the empirical risk (the uniform law of large numbers): $\sup_\alpha |R(\alpha, X_1^n) - R(\alpha)|$. Such a bound can be obtained from quantities that measure the size of a Glivenko-Cantelli class. For finite number of indices, the family size can be measured simply by its cardinality. For general function families, a well known quantity to measure the degree of uniform convergence is the *covering number* which can be be dated back to Kolmogrov [8, 9]. The idea is to discretize (which can depend on the data $X_1^n$) the parameter space into $N$ values $\alpha_1, \ldots, \alpha_N$ so that each $L(\alpha, \cdot)$ can be approximated by $L(\alpha_i, \cdot)$ for some $i$. We shall only describe a simplified version relevant for our purposes.

**Definition 2.1** *Let $B$ be a metric space with metric $\rho$. Given a norm p, observations $X_1^n = [x_1, \ldots, x_n]$, and vectors $f(\alpha, X_1^n) = [f(\alpha, x_1), \ldots, f(\alpha, x_n)] \in B^n$ parameterized by $\alpha$, the covering number in p-norm, denoted as $\mathcal{N}_p(f, \epsilon, X_1^n)$, is the minimum number of a collection of vectors $v_1, \ldots, v_m \in B^n$ such that $\forall \alpha, \exists v_i : \|\rho(f(\alpha, X_1^n), v_i)\|_p \leq n^{1/p} \epsilon$. We also denote $\mathcal{N}_p(f, \epsilon, n) = \max_{X_1^n} \mathcal{N}_p(f, \epsilon, X_1^n)$.*

Note that from the definition and the Jensen's inequality, we have $\mathcal{N}_p \leq \mathcal{N}_q$ for $p \leq q$. We will always assume the metric on $R$ to be $|x_1 - x_2|$ if not explicitly specified otherwise. The following theorem is due to Pollard [11]:

**Theorem 2.1 ([11])** $\forall n,\ \epsilon > 0$ *and distribution D.*

$$P[\sup_{\alpha} |R(\alpha, X_1^n) - R(\alpha)| > \epsilon] \leq 8E[\mathcal{N}_1(L, \epsilon/8, X_1^n)] \exp(\frac{-n\epsilon^2}{128M^2}),$$

*where* $M = \sup_{\alpha, x} L(\alpha, x) - \inf_{\alpha, x} L(\alpha, x)$, *and* $X_1^n = \{x_1, \ldots, x_n\}$ *are independently drawn from D.*

The constants in the above theorem can be improved for certain problems; see [4, 6, 15, 16] for related results. However, they yield very similar bounds. The result most relevant for this paper is a lemma in [3] where the 1-norm covering number is replaced by the $\infty$-norm covering number. The latter can be bounded by a scale-sensitive combinatorial dimension [1], which can be bounded from the 1-norm covering number if this covering number does not depend on $n$. These results can replace Theorem 2.1 to yield better estimates under certain circumstances.

Since Bartlett's lemma in [3] is only for binary loss functions, we shall give a generalization so that it is comparable to Theorem 2.1:

**Theorem 2.2** *Let* $f_1$ *and* $f_2$ *be two functions:* $R^n \to [0, 1]$ *such that* $|y_1 - y_2| \leq \gamma$ *implies* $f_1(y_1) \leq f_3(y_2) \leq f_2(y_1)$ *where* $f_3 : R^n \to [0, 1]$ *is a reference separating function, then*

$$P[\sup_{\alpha}[E_x f_1(L(\alpha, x)) - E_{X_1^n} f_2(L(\alpha, x))] > \epsilon] \leq 4E[\mathcal{N}_{\infty}(L, \gamma, X_1^n)] \exp(\frac{-n\epsilon^2}{32}).$$

Note that in the extreme case that some choice of $\alpha$ achieves perfect generalization: $E_x f_2(L(\alpha, x)) = 0$, and assume that our choices of $\alpha(X_1^n)$ always satisfy the condition $E_{X_1^n} f_2(L(\alpha, x)) = 0$, then better bounds can be obtained by using a refined version of the Chernoff bound.

## 3  Covering number bounds for linear systems

In this section, we present a few new bounds on covering numbers for the following form of real valued loss functions:

$$L(w, x) = x^T w = \sum_{i=1}^{d} x_i w_i. \tag{4}$$

As we shall see later, these bounds are relevant to the convergence properties of (1). Note that in order to apply Theorem 2.1, since $\mathcal{N}_1 \leq \mathcal{N}_2$, therefore it is sufficient to estimate $\mathcal{N}_2(L, \epsilon, n)$ for $\epsilon > 0$. It is clear that $\mathcal{N}_2(L, \epsilon, n)$ is not finite if no restrictions on $x$ and $w$ are imposed. Therefore in the following, we will assume that each $\|x_i\|_p$ is bounded, and study conditions of $\|w\|_q$ so that $\log \mathcal{N}(f, \epsilon, n)$ is independent or weakly dependent of $d$.

Our first result generalizes a theorem of Bartlett [3]. The original results is with $p = \infty$ and $q = 1$, and the related technique has also appeared in [10, 13]. The proof uses a lemma that is attributed to Maurey (cf. [2, 7]).

**Theorem 3.1** *If* $\|x_i\|_p \leq b$ *and* $\|w\|_q \leq a$, *where* $1/p + 1/q = 1$ *and* $2 \leq p \leq \infty$, *then*

$$\log_2 \mathcal{N}_2(L, \epsilon, n) \leq \lceil \frac{a^2 b^2}{\epsilon^2} \rceil \log_2(2d + 1).$$

The above bound on the covering number depends logarithmically on $d$, which is already quite weak (as compared to linear dependency on $d$ in the standard situation). However, the bound in Theorem 3.1 is not tight for $p < \infty$. For example, the following theorem improves the above bound for $p = 2$. Our technique of proof relies on the SVD decomposition [5] for matrices, which improves a similar result in [14] by a logarithmic factor.

**Theorem 3.2** *If $\|x_i\|_2 \le b$ and $\|w\|_2 \le a$, then*

$$\log_2 \mathcal{N}_2(L, \epsilon, n) \le \lceil \frac{2a^2 b^2}{\epsilon^2} \rceil \log_2(4a^2 b^2/\epsilon^2 + 1).$$

The next theorem shows that if $1/p + 1/q > 1$, then the 2-norm covering number is also independent of dimension.

**Theorem 3.3** *Let $L(w, x) = x^T w$. If $\|x_i\|_p \le b$ and $\|w\|_q \le a$, where $1 \le q \le 2$ and $\delta = 1/p + 1/q - 1 > 0$, then*

$$\log_2 \mathcal{N}_2(L, \epsilon, n) \le \lceil \frac{4a^2 b^2}{\epsilon^2} \rceil \log_2(2(2ab/\epsilon)^{1/\delta} + 1).$$

One consequence of this theorem is a potentially refined explanation for the boosting algorithm. In [13], the boosting algorithm has been analyzed by using a technique related to results in [3] which essentially rely on Theorem 3.1 with $p = \infty$. Unfortunately, the bound contains a logarithmic dependency on $d$ (in the most general case) which does not seem to fully explain the fact that in many cases the performance of the boosting algorithm keeps improving as $d$ increases. However, this seemingly mysterious behavior might be better understood from Theorem 3.3 under the assumption that the data is more restricted than simply being $\infty$-norm bounded. For example, when the contribution of the wrong predictions is bounded by a constant (or grow very slowly as $d$ increases), then we can regard its $p$-th norm bounded for some $p < \infty$. In this case, Theorem 3.3 implies dimensional independent generalization.

If we want to apply Theorem 2.2, then it is necessary to obtain bounds for infinity-norm covering numbers. The following theorem gives such bounds by using a result from online learning.

**Theorem 3.4** *If $\|x_i\|_p \le b$ and $\|w\|_q \le a$, where $2 \le p < \infty$ and $1/p + 1/q = 1$, then $\forall \epsilon > 0$,*

$$\log_2 \mathcal{N}_\infty(L, \epsilon, n) \le 36(p-1)\frac{a^2 b^2}{\epsilon^2} \log_2[2\lceil 4ab/\epsilon + 2\rceil n + 1].$$

In the case of $p = \infty$, an entropy condition can be used to obtain dimensional independent covering number bounds.

**Definition 3.1** *Let $\mu = [\mu_i]$ be a vector with positive entries such that $\|\mu\|_1 = 1$ (in this case, we call $\mu$ a distribution vector). Let $x = [x_i] \ne 0$ be a vector of the same length, then we define the weighted relative entropy of $x$ with respect to $\mu$ as:*

$$\text{entro}_\mu(x) = \sum_i |x_i| \ln \frac{|x_i|}{\mu_i \|x\|_1}.$$

**Theorem 3.5** *Given a distribution vector $\mu$, If $\|x_i\|_\infty \le b$ and $\|w\|_1 \le a$ and $\text{entro}_\mu(w) \le c$, where we assume that $w$ has non-negative entries, then $\forall \epsilon > 0$,*

$$\log_2 \mathcal{N}_\infty(L, \epsilon, n) \le \frac{36b^2(a^2 + ac)}{\epsilon^2} \log_2[2\lceil 4ab/\epsilon + 2\rceil n + 1].$$

Theorems in this section can be combined with Theorem 4.1 to form more complex covering number bounds for nonlinear compositions of linear functions.

## 4   Nonlinear extensions

Consider the following system:

$$L([\alpha, w], x) = f(g(\alpha, x) + w^T h(\alpha, x)), \tag{5}$$

where $x$ is the observation, and $[\alpha, w]$ is the parameter. We assume that $f$ is a nonlinear function with bounded total variation.

**Definition 4.1** *A function $f : R \to R$ is said to satisfy the Lipschitz condition with parameter $\gamma$ if $\forall x, y$: $|f(x) - f(y)| \le \gamma|x - y|$.*

**Definition 4.2** *The total variation of a function $f : R \to R$ is defined as*

$$\mathrm{TV}(f, x) = \sup_{x_0 < x_1 \cdots < x_\ell \le x} \sum_{i=1}^{\ell} |f(x_i) - f(x_{i-1})|.$$

*We also denote $\mathrm{TV}(f, \infty)$ as $\mathrm{TV}(f)$.*

**Theorem 4.1** *If $L([\alpha, w], x) = f(g(\alpha, x) + w^T h(\alpha, x))$, where $\mathrm{TV}(f) < \infty$ and $f$ is Lipschitz with parameter $\gamma$. Assume also that $w$ is a $d$-dimensional vector and $\|w\|_q \le c$, then $\forall \epsilon_1, \epsilon_2 > 0$, and $n > 2(d+1)$:*

$$\log_2 \mathcal{N}_r(L, \epsilon_1 + \epsilon_2, n) \le (d+1)\log_2[\frac{en}{d+1}\max(\lfloor\frac{\mathrm{TV}(f)}{2\epsilon_1}\rfloor, 1)] + \log_2 \mathcal{N}_r([g, h], \epsilon_2/\gamma, n),$$

*where the metric of $[g, h]$ is defined as $|g_1 - g_2| + c\|h_1 - h_2\|_p$ $(1/p + 1/q = 1)$.*

**Example 4.1** Consider classification by hyperplane: $L(w, x) = I(w^T x < 0)$ where $I$ is the set indicator function. Let $L'(w, x) = f_0(w^T x)$ be another loss function where

$$f_0(z) = \begin{cases} 1 & z < 0 \\ 1 - z & z \in [0, 1] \\ 0 & z > 1 \end{cases}.$$

Instead of using ERM for estimating parameter that minimizes the risk of $L$, consider the scheme of minimize empirical risk associated with $L'$, under the assumption that $\|x\|_2 \le b$ and constraint that $\|w\|_2 \le a$. Denote the estimated parameter by $w_n$. It follows from the covering number bounds and Theorem 2.1 that with probability of at least $1 - \eta$:

$$E_x I(w_n^T x \le 0) \le \inf_{\|w\|_2 \le a} E_x f_0(w^T x) + O(\sqrt{\frac{n^{1/2}ab\ln(nab + 2) + \ln\frac{1}{\eta}}{n}}).$$

If we apply a slight generalization of Theorem 2.2 and the covering number bound of Theorem 3.4, then with probability of at least $1 - \eta$:

$$E_x I(w_n^T x \le 0) \le E_{X_1^n} I(w_n^T x \le 2\gamma) + O(\sqrt{\frac{1}{n}(\frac{a^2 b^2}{\gamma^2}\ln(ab/\gamma + 2) + \ln n + \ln\frac{1}{\eta})})$$

for all $\gamma \in (0, 1]$. □

Bounds given in this paper can be applied to show that under appropriate regularization conditions and assumptions on the data, methods based on (1) lead to generalization performances of the form $\tilde{O}(1/\sqrt{n})$, where $\tilde{O}$ symbol (which is independent of $d$) is used to indicate that the hidden constant may include a polynomial dependency on $\log(n)$. It is also important to note that in certain cases, $\lambda$ will not appear (or it has a small influence on the convergence) in the constant of $\tilde{O}$, as being demonstrated by the example in the next section.

## 5  Asymptotic analysis

The convergence results in the previous sections are in the form of VC style convergence in probability, which has a combinatorial flavor. However, for problems with differentiable function families involving vector parameters, it is often convenient to derive precise asymptotic results using the differential structure.

Assume that the parameter $\alpha \in R^m$ in (2) is a vector and $L$ is a smooth function. Let $\alpha^*$ denote the optimal parameter; $\nabla_\alpha$ denote the derivative with respect to $\alpha$; and $\Psi(\alpha, x)$ denote $\nabla_\alpha L(\alpha, x)$. Assume that

$$V = \int \nabla_\alpha \Psi(\alpha^*, x)\, dP(x)$$

$$U = \int \Psi(\alpha^*, x)\Psi(\alpha^*, x)^T\, dP(x).$$

Then under certain regularity conditions, the asymptotic expected generalization error is given by

$$E\, R(\alpha_{\text{erm}}) = R(\alpha^*) + \frac{1}{2n}\text{tr}(V^{-1}U). \tag{6}$$

More generally, for any evaluation function $h(\alpha)$ such that $\nabla h(\alpha^*) = 0$:

$$E\, h(\alpha_{\text{erm}}) \approx h(\alpha^*) + \frac{1}{2n}\text{tr}(V^{-1}\nabla^2 h \cdot V^{-1}U), \tag{7}$$

where $\nabla^2 h$ is the Hessian matrix of $h$ at $\alpha^*$. Note that this approach assumes that the optimal solution is unique. These results are exact asymptotically and provide better bounds than those from the standard PAC analysis.

**Example 5.1** We would like to study a form of the support vector machine: Consider $L(\alpha, x) = f(\alpha^T x) + \frac{1}{2}\lambda\alpha^2$,

$$f(z) = \begin{cases} 1 - z & z \leq 1 \\ 0 & z > 1 \end{cases}.$$

Because of the discontinuity in the derivative of $f$, the asymptotic formula may not hold. However, if we make an assumption on the smoothness of the distribution $x$, then the expectation of the derivative over $x$ can still be smooth. In this case, the smoothness of $f$ itself is not crucial. Furthermore, in a separate report, we shall illustrate that similar small sample bounds without any assumption on the smoothness of the distribution can be obtained by using techniques related to asymptotic analysis.

Consider the optimal parameter $\alpha^*$ and let $S = \{x : \alpha^{*T}x \leq 1\}$. Note that $\lambda\alpha^* = E_{x \in S}x$, and $U = E_{x \in S}(x - E_{x \in S}x)(x - E_{x \in S}x)^T$. Assume that $\exists \gamma > 0$ s.t. $P(\alpha^{*T}x \leq \gamma) = 0$, then $V = \lambda I + B$ where $B$ is a positive semi-definite matrix. It follows that

$$\text{tr}(V^{-1}U) \leq \text{tr}(U)/\lambda \leq \frac{E_{x \in S}x^2}{E_{x \in S}\alpha^{*T}x}\|\alpha^*\|_2^2 \leq \sup \|x\|_2^2\|\alpha^*\|_2^2/\gamma.$$

Now, consider $\alpha_n$ obtained from observations $X_1^n = [x_1, \ldots, x_n]$ by minimizing empirical risk associated with loss function $L(\alpha, x)$, then

$$E_x L(\alpha_{emp}, x) \leq \inf_\alpha E_x L(\alpha, x) + \frac{1}{2\gamma n}\sup \|x\|_2^2\|\alpha^*\|_2^2$$

asymptotically. Let $\lambda \to 0$, this scheme becomes the optimal separating hyperplane [16]. This asymptotic bound is better than typical PAC bounds with fixed $\lambda$. $\square$

Note that although the bound obtained in the above example is very similar to the mistake bound for the perceptron online update algorithm, we may in practice obtain much better estimates from (6) by plugging in the empirical data.

# References

[1] N. Alon, S. Ben-David, N. Cesa-Bianchi, and D. Haussler. Scale-sensitive dimensions, uniform convergence, and learnability. *Journal of the ACM*, 44(4):615–631, 1997.

[2] A.R. Barron. Universal approximation bounds for superpositions of a sigmoidal function. *IEEE Transactions on Information Theory*, 39(3):930–945, 1993.

[3] P.L. Bartlett. The sample complexity of pattern classification with neural networks: the size of the weights is more important than the size of the network. *IEEE Transactions on Information Theory*, 44(2):525–536, 1998.

[4] R.M. Dudley. *A course on empirical processes*, volume 1097 of *Lecture Notes in Mathematics*. 1984.

[5] G.H. Golub and C.F. Van Loan. *Matrix computations*. Johns Hopkins University Press, Baltimore, MD, third edition, 1996.

[6] D. Haussler. Generalizing the PAC model: sample size bounds from metric dimension-based uniform convergence results. In *Proc. 30th IEEE Symposium on Foundations of Computer Science*, pages 40–45, 1989.

[7] Lee K. Jones. A simple lemma on greedy approximation in Hilbert space and convergence rates for projection pursuit regression and neural network training. *Ann. Statist.*, 20(1):608–613, 1992.

[8] A.N. Kolmogorov. Asymptotic characteristics of some completely bounded metric spaces. *Dokl. Akad. Nauk. SSSR*, 108:585–589, 1956.

[9] A.N. Kolmogorov and V.M. Tihomirov. $\epsilon$-entropy and $\epsilon$-capacity of sets in functional spaces. *Amer. Math. Soc. Transl.*, 17(2):277–364, 1961.

[10] Wee Sun Lee, P.L. Bartlett, and R.C. Williamson. Efficient agnostic learning of neural networks with bounded fan-in. *IEEE Transactions on Information Theory*, 42(6):2118–2132, 1996.

[11] D. Pollard. *Convergence of stochastic processes*. Springer-Verlag, New York, 1984.

[12] N. Sauer. On the density of families of sets. *Journal of Combinatorial Theory (Series A)*, 13:145–147, 1972.

[13] Robert E. Schapire, Yoav Freund, Peter Bartlett, and Wee Sun Lee. Boosting the margin: a new explanation for the effectiveness of voting methods. *Ann. Statist.*, 26(5):1651–1686, 1998.

[14] J. Shawe-Taylor, P.L. Bartlett, R.C. Williamson, and M. Anthony. Structural risk minimization over data-dependent hierarchies. *IEEE Trans. Inf. Theory*, 44(5):1926–1940, 1998.

[15] V.N. Vapnik. *Estimation of dependences based on empirical data*. Springer-Verlag, New York, 1982. Translated from the Russian by Samuel Kotz.

[16] V.N. Vapnik. *The nature of statistical learning theory*. Springer-Verlag, New York, 1995.

[17] V.N. Vapnik and A.J. Chervonenkis. On the uniform convergence of relative frequencies of events to their probabilities. *Theory of Probability and Applications*, 16:264–280, 1971.

[18] Tong Zhang. Analysis of regularized linear functions for classification problems. Technical Report RC-21572, IBM, 1999.
